# Semi-Crowdsourced Clustering: Generalizing Crowd Labeling by Robust Distance Metric Learning

**Jinfeng Yi[†], Rong Jin[†], Anil K. Jain[†], Shaili Jain[♮], Tianbao Yang[‡]**
[†]Michigan State University, East Lansing, MI 48824, USA
[♮]Yale University, New Haven, CT 06520, USA
[‡]Machine Learning Lab, GE Global Research, San Ramon, CA 94583, USA
{yijinfen, rongjin, jain}@cse.msu.edu, shaili.jain@yale.edu, tyang@ge.com

## Abstract

One of the main challenges in data clustering is to define an appropriate similarity measure between two objects. Crowdclustering addresses this challenge by defining the pairwise similarity based on the manual annotations obtained through crowdsourcing. Despite its encouraging results, a key limitation of crowdclustering is that it can only cluster objects when their manual annotations are available. To address this limitation, we propose a new approach for clustering, called *semi-crowdsourced clustering* that effectively combines the low-level features of objects with the manual annotations of a subset of the objects obtained via crowdsourcing. The key idea is to learn an appropriate similarity measure, based on the low-level features of objects and from the manual annotations of only a small portion of the data to be clustered. One difficulty in learning the pairwise similarity measure is that there is a significant amount of noise and inter-worker variations in the manual annotations obtained via crowdsourcing. We address this difficulty by developing a metric learning algorithm based on the matrix completion method. Our empirical study with two real-world image data sets shows that the proposed algorithm outperforms state-of-the-art distance metric learning algorithms in both clustering accuracy and computational efficiency.

## 1 Introduction

Crowdsourcing provides an easy and relatively inexpensive way to utilize human capabilities to solve difficult computational learning problems (e.g. image annotation in ESP game [17]). It divides a large task into a number of small-scale tasks, often referred to as *Human Intelligence Tasks* (HITs), and asks a human worker to solve each individual HIT. It then combines the partial solutions obtained from individual HITs to form the final solution. In the past, crowdsourcing has been explored for a number of machine learning tasks (e.g., classification and clustering) [21, 10, 19].

Crowdclustering [10] exploits the crowdsourcing paradigm for data clustering. The key idea is to first obtain manual annotations of objects through crowdsourcing. The annotations can either be in the form of grouping objects based on their perceived similarities [10] or the keyword assignments to individual objects (e.g., images) by human workers [25]. A pairwise similarity matrix is then computed from the acquired annotations, and is used to cluster objects. Unlike the conventional clustering techniques where the similarity measure is defined based on the features of objects, in crowdclustering, the pairwise similarities are derived from the manual annotations, which better capture the underlying inter-object similarity. Studies [10] have shown that crowdclustering performs significantly better than the conventional clustering methods, given a sufficiently large number of manual annotations for all the objects to be clustered.

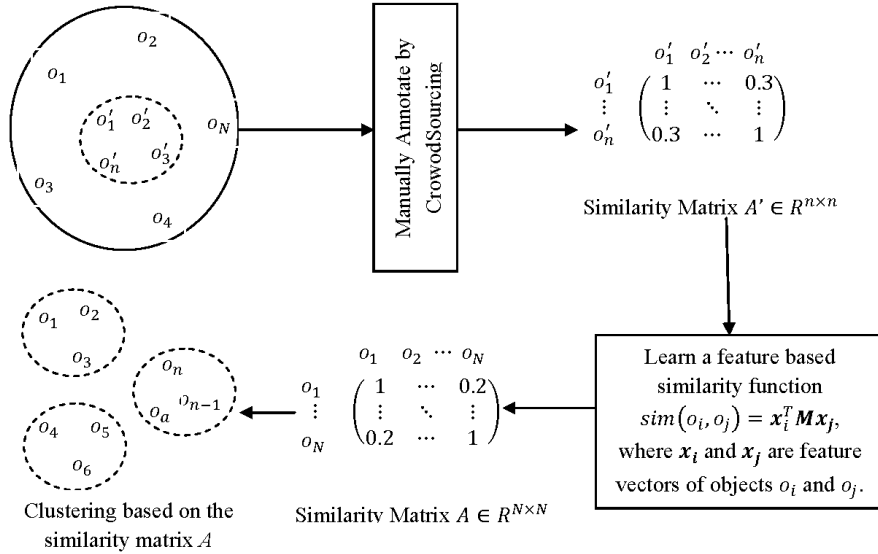

Figure 1: The proposed framework for semi-crowdsourced clustering. The given $N$ objects $(o_1, o_2, \ldots, o_N)$ need to be clustered, but only a small subset of the $N$ objects $(o'_1, o'_2, \cdots o'_n)$ have been annotated by crowdsourcing, $n \ll N$.

Despite the encouraging results obtained via crowdclustering, a main shortcoming of crowdclustering is that it can only cluster objects for which manual annotations are available, significantly limiting its application to large scale clustering problems. For instance, when clustering hundreds of thousands of objects, it is not feasible to have each object manually annotated by multiple workers. To address this limitation, we study the problem of **semi-crowdsourced clustering**, where given the annotations obtained through crowdsourcing for a small subset of the objects, the objective is to cluster the entire collection of objects. Figure 1 depicts the proposed framework. Given a set of $N$ objects to be clustered, the objective is to learn a pairwise similarity measure from the crowdsourced labels of $n$ objects ($n \ll N$) and the object feature vector $x$. Note that the available crowdclustering algorithms [10, 25] expect that all $N$ objects be labeled by crowdsourcing.

The key to semi-crowdsourced clustering is to define an appropriate similarity measure for the subset of objects that do not have manual annotations (i.e., $N - n$ objects). To this end, we propose to learn a similarity function, based on the object features, from the pairwise similarities derived from the manual annotations for the subset of $n$ objects; we then apply the learned similarity function to compute the similarity between any two objects, and perform data clustering based on the computed similarities. In this study, for computational simplicity, we restrict ourselves to a linear similarity function, i.e. given two objects $o_i$ and $o_j$ and their feature representation $\mathbf{x}_i$ and $\mathbf{x}_j$, respectively, their similarity $sim(O_i, O_j)$ is given by $sim(O_i, O_j) = \mathbf{x}_i^\top M \mathbf{x}_j$, where $M \succeq 0$ is the learned distance metric.

Learning a linear similarity function from given pairwise similarities (sometimes referred to as pairwise constraints when similarities are binary) is known as *distance metric learning*, which has been studied extensively in the literature [24]. The key challenge of distance metric learning in semi-crowdsourced clustering arises due to the noise in the pairwise similarities obtained from manual annotations. According to [25], large disagreements are often observed among human workers in specifying pairwise similarities. As a result, pairwise similarities based on the majority vote among human workers often disagree with the true cluster assignments of objects. As an example, the authors in [25] show that for the Scenes data set [8], more than $80\%$ of the pairwise labels obtained from human workers are inconsistent with the true cluster assignment. This large noise in the pairwise similarities due to crowdsourcing could seriously misguide the distance metric learning and lead to a poor prediction performance, as already demonstrated in [12] as well as in our empirical study.

We propose a metric learning algorithm that explicitly addresses the presence of noise in pairwise similarities obtained via crowdsourcing. The proposed algorithm uses the matrix completion technique [3] to rectify the noisy pairwise similarities, and regression analysis to efficiently learn a

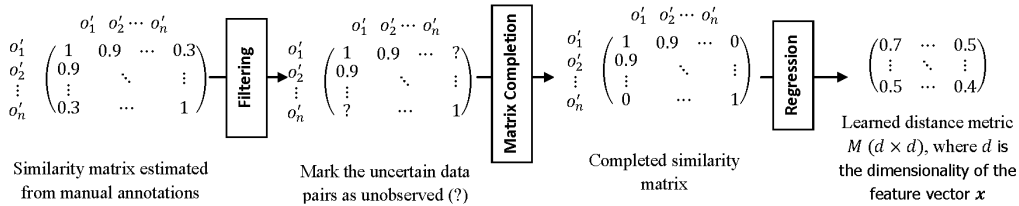

Figure 2: The proposed framework of learning a distance metric from noisy manual annotations

distance metric from the restored pairwise similarities. More specifically, the proposed algorithm for clustering $N$ objects consists of three components: (i) filtering noisy pairwise similarities for $n$ objects by only keeping object pairs whose pairwise similarities are agreed by many workers (not majority of the workers). The result of the filtering step is a partially observed $n \times n$ similarity matrix ($n \ll N$) with most of its entries removed/unobserved; (ii) recovering the $n \times n$ similarity matrix from the partially observed entries by using the matrix completion algorithm; (iii) applying a regression algorithm to learn a distance metric from the recovered similarity matrix, and clustering the $N \times N$ pairwise similarities based on the learned distance metric. Figure 2 shows the basic steps of the proposed algorithm.

Compared to the existing approaches of distance metric learning [24], the proposed algorithm has the following three advantages: (i) by exploring the matrix completion technique, the proposed algorithm is robust to a large amount of noise in the pairwise similarities; (ii) by utilizing regression analysis, the proposed algorithm is computationally efficient and does not have to handle the positive semi-definite constraint, a key computational bottleneck for most distance metric learning algorithms; (iii) the learned distance metric, with high probability, is close to the optimal metric learned from the perfect or true similarities (i.e. similarity of 1 when two objects are in the same cluster and 0, otherwise) for arbitrarily large $n$.

We finally note that in addition to distance metric learning, both kernel learning [16] and constrained clustering [2] can be applied to generalize the information in the manual annotations acquired by crowdsourcing. In this work, we focus on distance metric learning. The related work, as well as the discussion on exploring kernel learning and constrained clustering techniques for semi-crowdsourced clustering can be found in Section 4.

## 2 Semi-Crowdsourced Clustering by Robust Distance Metric Learning

We first present the problem and a general framework for semi-crowdsourced clustering. We then describe the proposed algorithm for learning distance metric from a small set of noisy pairwise similarities that are derived from manual annotations.

### 2.1 Problem Definition and Framework

Let $\mathcal{D} = \{O_1, \ldots, O_N\}$ be the set of $N$ objects to be clustered, and let $X = (\mathbf{x}_1, \ldots, \mathbf{x}_N)$ be their feature representation, where $\mathbf{x}_i \in \mathbb{R}^d$ is a vector of $d$ dimensions. We randomly sample a subset of $n \ll N$ objects from the collection $\mathcal{D}$, denoted by $\widehat{\mathcal{D}} = \{\widehat{O}_1, \ldots, \widehat{O}_n\}$, and obtain their manual annotations by crowdsourcing. Let $m$ be the number of HITs used by crowdsourcing. Given the manual annotations collected from the $k$-th HIT, we define a similarity matrix $A^k \in \mathbb{R}^{n \times n}$ such that $A^k_{i,j} = 1$ if objects $\widehat{O}_i$ and $\widehat{O}_j$ share common annotations (i.e. share common annotated keywords or assigned to the same cluster by the worker), zero if they don't, and $-1$ if either of the two objects is not annotated by the $k$th HIT (i.e. unlabeled pair). Note that we only consider a binary similarity measure in this study because our goal is to *perfectly* reconstruct the ideal pairwise similarities based on the true cluster assignments (i.e. 1 when both objects are assigned to the same cluster and zero, otherwise). The objective of semi-crowdsourced clustering is to cluster all the $N$ objects in $\mathcal{D}$ based on the features in $X$ and the $m \times m$ similarity matrices $\{A^k\}_{k=1}^m$ for the objects in $\widehat{\mathcal{D}}$. Throughout this paper, we assume that the number of clusters, denoted by $r$, is given a priori [1].

To generalize the pairwise similarities from the subset $\widehat{\mathcal{D}}$ to the entire collection of objects $\mathcal{D}$, we propose to first learn a distance metric from the similarity matrices $\{A^k\}_{k=1}^m$, and then compute the pairwise similarity for all the $N$ objects in $\mathcal{D}$ using the learned distance metric. The challenge is how to learn an appropriate distance metric from a set of similarity matrices $\{A^k\}_{k=1}^m$. A straightforward approach is to combine multiple similarity matrices into a single similarity matrix by computing their average. More specifically, let $\widetilde{A} \in \mathbb{R}^{n \times n}$ be the average similarity matrix. We have

$$\widetilde{A}_{i,j} = \frac{1}{\sum_{k=1}^m I(A_{i,j}^k \geq 0)} \sum_{k=1}^m I(A_{i,j}^k \geq 0) A_{i,j}$$

where $A_{i,j}^k < 0$ indicates that the pair $(\widehat{O}_i, \widehat{O}_j)$ is not labeled by the $k$th HIT (i.e. either object $\widehat{O}_i$ or $\widehat{O}_j$ is not annotated by the $k$th worker) and $I(z)$ is an indicator function that outputs $1$ when $z$ is true and zero, otherwise. We then learn a distance metric $M$ from $\widetilde{A}$. The main problem with this simple strategy is that due to the large disagreements among workers in determining the pairwise similarities, the average similarities do not correlate well with the true cluster assignments. In the next subsection, we develop an efficient and robust algorithm that learns a distance metric from a set of noisy similarity matrices.

## 2.2 Learning a Distance Metric from a Set of Noisy Similarity Matrices

As illustrated in Figure 2, the proposed algorithm consists of three steps, i.e. filtering step, matrix completion step and distance metric learning step. For the first two steps, namely the data prepro-cessing steps, we follow the idea proposed in [25].

**Filtering step.** To filter out the uncertain object pairs, we introduce two thresholds $d_0$ and $d_1 (1 \geq d_1 > d_0 \geq 0)$ into the average similarity matrix $\tilde{A}$. Since any similarity measure smaller than $d_0$ indicates that most workers put the corresponding object pair into different clusters, we simply set it as 0. Similarly, we set the similarity measure larger than $d_1$ as 1. For object pairs with similarity measure in the range between $d_0$ and and $d_1$, they are treated as uncertain object pairs and are discarded (i.e. marked as unobserved) from the similarity matrix. The resulting partially observed similarity matrix $A$ is given by

$$A_{i,j} = \begin{cases} 1 & \tilde{A}_{i,j} \in [d_1, 1] \\ 0 & \tilde{A}_{i,j} \in [0, d_0] \\ \text{unobserved} & \text{Otherwise} \end{cases} \tag{1}$$

We also define $\Delta$ as the set of observed entries in $A_{i,j}$

$$\Delta = \{(i,j) \in [N] \times [N] : \tilde{A}_{ij} \geq 0, \tilde{A}_{ij} \notin (d_0, d_1)\}$$

**Matrix completion step.** Since $A$ is constructed from the partial clustering results generated by different workers, we expect some of the binary similarity measures in $A$ to be incorrect. We in-troduce the matrix $E \in \mathbb{R}^{n \times n}$ to capture the incorrect entries in $A$. If $A^*$ is the perfect similarity matrix, we have $\mathcal{P}_\Delta(A^* + E) = \mathcal{P}_\Delta(A)$, where $\mathcal{P}_\Delta$ outputs a matrix with $[\mathcal{P}_\Delta(B)]_{i,j} = B_{i,j}$ if $(i,j) \in \Delta$ and zero, otherwise. With appropriately chosen thresholds $d_0$ and $d_1$, we expect most of the observed entries in $A$ to be correct and as a result, $E$ to be a sparse matrix. To reconstruct the perfect similarity matrix $A^*$ from $A$, following the matrix completion theory [3], we solve the following optimization problem

$$\min_{\widehat{A}, E} |\widehat{A}|_* + C|E|_1 \text{ s. t. } \mathcal{P}_\Delta(\widehat{A} + E) = \mathcal{P}_\Delta(A), \tag{2}$$

where $|A|_*$ is the nuclear norm of matrix $A$ and $|E|_1 = \sum_{i,j} |E_{i,j}|$ is the $\ell_1$ norm of $E$. Using the facts that $E$ is a sparse matrix and $\widehat{A}$ is of low rank [14], under the two assumptions made in [25], with a high probability, we have $A^* = \widehat{A}$, where $\widehat{A}$ is the optimal solution for (2). For completeness, we include in the supplementary document the theoretical result for the problem in (2)

**Distance metric learning step.** This step learns a distance metric from the completed similarity matrix $\widehat{A}$. A common problem shared by most distance metric learning algorithms is their high computational cost due to the constraint that a distance metric has to be positive semi-definite. In this study, we develop an efficient algorithm for distance metric learning that does not have to deal with

the positive semi-definite constraint. Our algorithm is based on the key observation that with a high probability, the completed similarity matrix $\widehat{A}$ is positive semi-definite. This is because according to Theorem 1 of [25], with a probability at least $1 - n^{-3}$, $\widehat{A} = YY^\top$, where $Y \in \{0, 1\}^{n \times r}$ is the true cluster assignment. This property guarantees the resulting distance metric to be positive semi-definite.

The proposed distance metric learning algorithm is based on a standard regression algorithm [15]. Given the similarity matrix $\widehat{A}$, the optimal distance metric $M$ is given by a regression problem

$$\min_{M \in \mathbb{R}^{d \times d}} \quad \widehat{\mathcal{L}}(M) = \sum_{i,j=1}^n (\widehat{\mathbf{x}}_i^\top M \widehat{\mathbf{x}}_j - \widehat{A}_{i,j})^2 = |\widehat{X}^\top M \widehat{X} - \widehat{A}|_F^2 \tag{3}$$

where $\widehat{\mathbf{x}}_i$ is the feature vector for the sampled object $\widehat{O}_i$ and $\widehat{X} = (\widehat{\mathbf{x}}_1, \ldots, \widehat{\mathbf{x}}_n)$. The optimal solution to (3), denoted by $\widehat{M}$, is given by

$$\widehat{M} = (\widehat{X}\widehat{X}^\top)^{-1} \widehat{X} \widehat{A} \widehat{X}^\top (\widehat{X}\widehat{X}^\top)^{-1} \tag{4}$$

where $Z^{-1}$ is pseudo inverse of $Z$. It is straightforward to verify $\widehat{M} \succeq 0$ if $\widehat{A} \succeq 0$.

Directly using the solution in (4) could result in the overfitting of similarity matrix $\widehat{A}$ because of the potential singularity of $\widehat{X}\widehat{X}^\top$. We address this challenge by a smoothing technique, i.e.

$$\widehat{M}_s = (\widehat{X}\widehat{X}^\top + \lambda m I)^{-1} \widehat{X} \widehat{A} \widehat{X}^\top (\widehat{X}\widehat{X}^\top + \lambda m I)^{-1} \tag{5}$$

where $I$ is the identity matrix of size $d \times d$ and $\lambda > 0$ is a smoothing parameter used to address the overfitting and the curse of dimensionality. Note that the computation in (5) can be simplified by expressing $\widehat{M}_s$ in terms of the singular values and singular vectors of $\widehat{X}$. We omit the details due to the space constraints.

We now state the theoretical property of $\widehat{M}_s$. Let $A(O_i, O_j)$ be the perfect similarity that outputs 1 when $O_i$ and $O_j$ belong to the same cluster and zero, otherwise. It is straightforward to see that $A(O_i, O_j) = \mathbf{y}_i^\top \mathbf{y}_j$, where $\mathbf{y}_i \in \{0, 1\}^r$ is the cluster assignment for object $O_i$. To learn an ideal distance metric from the perfect similarity measure $A(O_i, O_j)$, we generalize the regression problem in (3) as follows

$$\min_{M \in \mathbb{R}^{d \times d}} \quad \mathcal{L}(M) = \mathrm{E}_{\mathbf{x}_i, \mathbf{x}_j} \left[ (\mathbf{x}_i^\top M \mathbf{x}_j - A(O_i, O_j))^2 \right] \tag{6}$$

The solution to (6) is given by $M = C_X^{-1} BB^\top C_X^{-1}$, where $C_X = \mathrm{E}_{\mathbf{x}_i}[\mathbf{x}_i \mathbf{x}_i^\top]$ and $B = \mathrm{E}_{\mathbf{x}_i}[\mathbf{x}_i \mathbf{y}_i^\top]$. Let $M_s$ be the smoothed version of the ideal distance metric $M$, i.e. $M = (C_X + \lambda I)^{-1} BB^\top (C_X + \lambda I)^{-1}$. The following theorem shows that with a high probability, the difference between $\widehat{M}_s$ and $M_s$ is small if both $\lambda$ and $n$ are not too small.

**Theorem 1.** *Assume $|\mathbf{x}|_2 \leq 1$ for the feature representation of any object. Assume the conditions in Theorem 1 of [25] hold. Then, with a probability $1 - 3n^{-3}$, we have*

$$|M_s - \widehat{M}_s|_2 = O \left( \frac{\ln n}{\lambda^2 \sqrt{n}} \right)$$

*where $|Z|_2$ stands for the spectral norm of matrix $Z$.*

The detailed proof can be found in the supplementary materials. Given the learned distance metric $\widehat{M}_s$, we construct a similarity matrix $S = X^\top \widehat{M}_s X$ and then apply a spectral clustering algorithm [18] to compute the final data partition for $N$ objects.

## 3 Experiments

In this section, we demonstrate empirically that the proposed semi-crowdsourced clustering algorithm is both effective and efficient.

### 3.1 Data Sets, Baselines, and Parameter Settings

**Data Sets.** Two real-world image data sets are used in our experiments: (i) *ImageNet data set* is a subset of the larger ImageNet database [6]. The subset contains $6,408$ images belonging to 7 categories: *tractor*, *horse cart*, *bench*, *blackberry*, *violin*, *saxophone*, and *hammer*. (ii) *PASCAL 07 data set* is a subset of the PASCAL Visual Object Classes Challenge 2007 database [7]. The subset contains $2,989$ images belonging to five classes: *car*, *dog*, *chair*, *cat* and *bird*. We choose these specific image categories because they yield relatively low classification performance in ImageNet competition and PASCAL VOC Challenge, indicating that it could be difficult to cluster these images using low level features without side information. The image features for these datasets were downloaded from the homepages of the ImageNet database [2] and the research group of Learning and Recognition in Vision (LEAR) [3], respectively.

To perform crowdlabeling, we follow [25], and ask human workers to annotate images with keywords of their choice in each HIT. A total of 249 and 332 workers were employed using the Amazon's Mechanical Turk [13] to annotate images from ImageNet and PASCAL datasets, respectively. On average, each image is annotated by five different workers, with three keywords from each individual worker. For every HIT, the pairwise similarity between two images (i.e. $A_{i,j}^k$ used in Section 2.1) is set to 1 if the two images share at least one common annotated keyword and zero, otherwise [4].

**Baselines.** Two baseline methods are used as reference points in our study: (a) the **Base** method that clusters images directly using image features without distance metric learning, and (b) the **Raw** method that runs the proposed algorithm against the average similarity matrix $\widetilde{A}$ without filtering and matrix completion steps. The comparison to the **Base** method allows us to examine the effect of distance metric learning in semi-crowdsourced clustering, and the comparison to the **Raw** method reveals the effect of filtering and matrix completion steps in distance metric learning.

We compare the proposed algorithm for distance metric learning to the following five state-of-the-art distance metric learning algorithms: (a) **GDM**, the global distance metric learning algorithm [23], (b) **RCA**, the relevant component analysis [1], (c) **DCA**, the discriminative component analysis [11], (d) **ITML**, the information theoretic metric learning algorithm [5], and (e) **LMNN**, the large margin nearest neighbor classifier [20]. Some of the other state-of-the-art distance metric learning algorithms (e.g. the neighborhood components analysis (NCA) [9]) were excluded from the comparison because they can only work with class assignments, instead of pairwise similarities, and therefore are not applicable in our case. The code for the baseline algorithms was provided by their respective authors (In LMNN, Principal Component Analysis (PCA) is used at first to reduce the data to lower dimensions). For a fair comparison, all distance metric learning algorithms are applied to the pairwise constraints derived from $\widehat{A}$, the $n \times n$ pairwise similarity matrix reconstructed by the matrix completion algorithm. We refer to the proposed distance metric learning algorithm as **Regression based Distance Metric Learning**, or **RDML** for short, and the proposed semi-crowdsourced clustering algorithm as **Semi-Crowd**.

**Parameter Settings.** Two criteria are used in determining the values for $d_0$ and $d_1$ in (1). First, $d_0$ ($d_1$) should be small (large) enough to ensure that most of the retained pairwise similarities are consistent with the cluster assignments. Second, $d_0$ ($d_1$) should be large (small) enough to obtain a sufficient number of observed entries in the partially observed matrix $A$. For both data sets, we set $d_0$ to 0 and $d_1$ to 0.8. We follow the heuristic proposed in [25] to determine the parameter $C$ in (2), which is selected to generate balanced clustering results. Parameter $\lambda$ in (5) is set to 1. We varied $\lambda$ from 0.5 to 5 and found that the clustering results essentially remain unchanged.

**Evaluation.** Normalized mutual information (NMI for short) [4] is used to measure the coherence between the inferred clustering and the ground truth categorization. The number of sampled images is varied from 100, 300, 600 to $1,000$. All the experiments are performed on a PC with Intel Xeon 2.40 GHz processor and 16.0 GB of main memory. Each experiment is repeated five times, and the performance averaged over the five trials is reported.

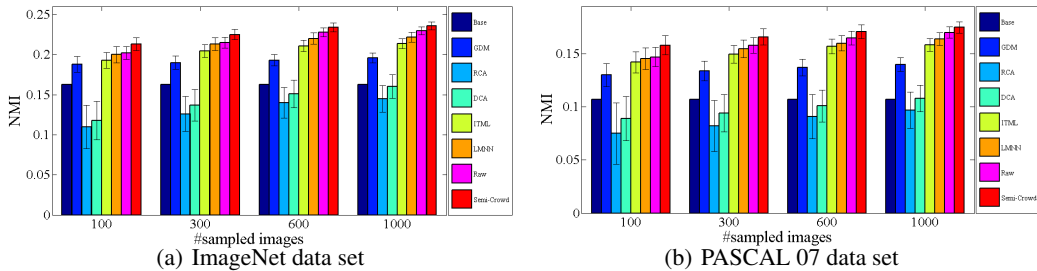

| (a) ImageNet data set | (b) PASCAL 07 data set |

Figure 3: NMI vs. no. of sampled images ($n$) used in crowdlabeling.

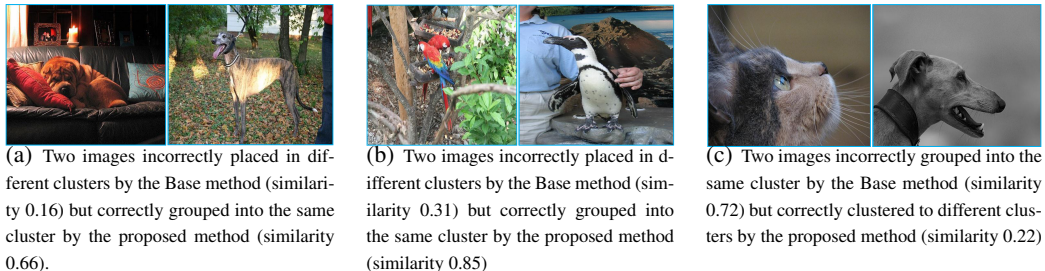

(a) Two images incorrectly placed in different clusters by the Base method (similarity 0.16) but correctly grouped into the same cluster by the proposed method (similarity 0.66).

(b) Two images incorrectly placed in different clusters by the Base method (similarity 0.31) but correctly grouped into the same cluster by the proposed method (similarity 0.85)

(c) Two images incorrectly grouped into the same cluster by the Base method (similarity 0.72) but correctly clustered to different clusters by the proposed method (similarity 0.22)

Figure 4: Sample image pairs that are incorrectly clustered by the Base method but correctly clustered by the proposed method (the similarity of our method is based on the normalized distance metric $\widehat{M_s}$).

## 3.2 Experimental Results

First, we examine the effect of distance metric learning algorithm on semi-crowdsourced clustering. Figure 3 compares the clustering performance with six different metric learning algorithms with that of the Base method that does not learn a distance metric. We observed that four of the distance metric learning algorithms (i.e. GDM, ITML, LMNN and the proposed RDML) outperform the Base method, while RCA and DCA fail to improve the clustering performance of Base. We conjecture that the failure of RCA and DCA methods is due to their sensitivity to the noisy pairwise similarities. In fact, RCA and DCA can yield better performance than the Base method if all the pairwise similarities are consistent with the cluster assignments. Compared to all the baseline distance metric learning algorithms, **RDML**, the proposed distance metric learning algorithm, yields the best clustering results for both the data sets and for all values of $n$ (i.e. the number of annotated images) considered here. Furthermore, the performance of **RDML** gradually stabilizes as the number of sampled images increases. This is consistent with our theoretical analysis in Theorem 1, and implies that only a modest number of annotated images is needed by the proposed algorithm to learn an appropriate distance metric. This observation is particularly useful for crowdclustering as it is expensive to reliably label a very large number of images. Figure 4 shows some example image pairs for which the Base method fails to make correct cluster assignments, but the proposed **RDML** method successfully corrects these mistakes with the learned distance metric.

Our next experiment evaluates the impact of filtering and matrix completion steps. In Figure 3, we compare the clustering results of the proposed algorithm for semi-crowdsourced clustering (i.e. Filtering+Matrix-Completion+RDML) to the Raw method that runs the proposed distance metric algorithm RDML without the filtering and matrix completion steps. Based on these experiments, we can make the following observations: (i) the proposed distance metric learning algorithms performs better than the Raw method, particularly when the number of annotated images is small; (ii) the gap between the proposed semi-crowdsourced clustering method and the Raw method decreases as the sample size increases. These results indicate the importance of filtering and matrix completion steps for the crowdsourced data in semi-crowdsourced clustering. Finally, it is interesting to observe that the Raw method still outperforms all the baseline methods, which further verifies the effectiveness of the proposed algorithm for distance metric learning.

Finally, we evaluate the computational efficiency of the proposed distance metric learning algorithm. Table 1 shows that the proposed distance metric learning algorithm is significantly more efficient than the baseline approaches evaluated here. The last row of Table 1 indicates the run time for the

Table 1: CPU time (in seconds) for learning the distance metrics.

| CPU time (s) | ImageNet Data Set | | | | PASCAL 07 Data Set | | | |
|---|---|---|---|---|---|---|---|---|
| Sample sizes ($n$) | 100 | 300 | 600 | 1,000 | 100 | 300 | 600 | 1,000 |
| RDML (proposed) | 4.2 | 6.3 | 8.0 | 11.2 | 27.4 | 34.2 | 41.7 | 47.3 |
| GDM [23] | 11384 | 14706 | 18140 | 25155 | 26346 | 36795 | 44237 | 53468 |
| LMNN [20] | 59.8 | 157 | 330 | 629 | 55.1 | 124 | 277 | 527 |
| ITML [5] | 2128 | 2376 | 2692 | 3081 | 5311 | 5721 | 6104 | 6653 |
| DCA [11] | 8.5 | 9.2 | 14.5 | 20.7 | 51.2 | 64.1 | 72.7 | 82.3 |
| RCA [1] | 9.7 | 13.5 | 18.6 | 23.6 | 71.4 | 92.7 | 103 | 122 |
| Matrix Completion | 12.4 | 74.2 | 536 | 1916 | 12.8 | 86.6 | 615 | 1873 |

matrix completion step. Since all the distance metric learning algorithms are applied to the similarity matrix recovered by the matrix completion algorithm, the computational cost of matrix completion is shared by all distance metric learning algorithms used in our evaluation. We observe that the matrix completion step, particularly for large sample sizes, is computationally demanding, a problem that will be investigated in our future work.

## 4   Related Work and Discussion

Crowdclustering was first proposed in [10]. It divided the task of clustering a collection of images into a number of human intelligence tasks (or HITs). In each HIT, a small subset of images are randomly sampled from the collection, and a worker is asked to cluster the subset of images into multiple groups. By using a large number of HITs, the authors ensure that every image in the collection is included in at least one HIT. In [25], the authors extend the definition of HITs for crowdclustering by asking workers to annotate images by keywords and then derive pairwise similarities between images based on the commonality of annotated keywords. A major limitation of both these studies, as pointed out earlier, is that they can only cluster images that have been manually annotated. Although the matrix completion technique was first proposed for crowdclustering in [25], it had a different goal from this work. In [25], matrix completion was used to estimate the similarity matrix, while the proposed approach uses matrix completion to estimate a distance metric, so that crowdsourced labels can be generalized to cluster those images which were not annotated during crowdsourcing.

Our work is closely related to distance metric learning that learns a distance metric consistent with a given subset of pairwise similarities/constraints [24]. Although many studies on distance metric learning have been reported, only a few address the challenge of learning a reliable distance metric from noisy pairwise constraints [12, 22]. One limitation of these earlier studies is that they can only work with a relatively small number (typically less than 30%) of noisy pairwise constraints. In contrast, in semi-crowdsourced clustering, we expect that a significantly larger percentage of pairwise similarities are inconsistent with the true cluster assignments (as many as 80% [25]).

One limitation of distance metric learning is that it is restricted to a linear similarity function. Kernel learning generalizes distance metric learning to a nonlinear similarity function by mapping each data point to a high dimensional space through a kernel function [16]. We plan to learn a kernel based similarity function from a subset of manually annotated objects. Besides distance metric learning, an alternative approach to incorporate the manual annotations into the clustering process is constrained clustering (or semi-supervised clustering) [2]. Compared to distance metric learning, constrained clustering can be computationally more expensive. Unlike distance metric learning that learns a distance metric from pairwise constraints only once and applies the learned distance metric to cluster any set of objects, a constrained clustering algorithm has to be rerun whenever a new set of objects needs to be clustered. To exploit the strength of constrained clustering algorithms, we plan to explore hybrid approaches that effectively combine distance metric learning with constrained clustering approaches for more accurate and efficient semi-crowdsourced clustering.

### Acknowledgments

This work was supported in part by National Science Foundation (IIS-0643494) and Office of Navy Research (Award nos. N00014-12-1-0431, N00014-11-1-0100, N00014-12-1-0522, and N00014-09-1-0663).

## Footnotes

[1]We may relax this requirement by estimating the number of clusters via some heuristic, e.g. considering the number of clusters as the rank of the completed matrix $A$.

[2] http://www.image-net.org/download-features

[3] http://lear.inrialpes.fr/people/guillaumin/data.php

[4] We tried several other similarity measures (e.g. cosine similarity measure and tf.idf weighting) and found that none of them yielded better performance than the simple similarity measure used in this work

# References

[1] Aharon Bar-Hillel, Tomer Hertz, Noam Shental, and Daphna Weinshall. Learning a Mahalanobis metric from equivalence constraints. *JMLR*, 2005.

[2] Sugato Basu, Ian Davidson, and Kiri Wagstaff. *Constrained Clustering: Advances in Algorithms, Theory, and Applications*. Chapman & Hall/CRC, 2008.

[3] Emmanuel J. Candès and Terence Tao. The power of convex relaxation: near-optimal matrix completion. *IEEE Transactions on Information Theory*, 56(5):2053–2080, 2010.

[4] Thomas M. Cover and Joy A. Thomas. *Elements of Information Theory (2nd ed.)*. Wiley, 2006.

[5] J.V. Davis, B. Kulis, P. Jain, S. Sra, and I.S. Dhillon. Information-theoretic metric learning. In *ICML*, pages 209–216, 2007.

[6] J. Deng, W. Dong, R. Socher, L.J. Li, K. Li, and L. Fei-Fei. Imagenet: A large-scale hierarchical image database. In *CVPR*, 2009.

[7] M. Everingham, L. Van Gool, C. K. I. Williams, J. Winn, and A. Zisserman. The PASCAL Visual Object Classes Challenge 2007 (VOC2007) Results. http://www.pascal-network.org/challenges/VOC/voc2007/workshop/index.html.

[8] L. Fei-Fei and P. Perona. A bayesian hierarchical model for learning natural scene categories. In *CVPR*, pages 524–531, 2005.

[9] Jacob Goldberger, Sam T. Roweis, Geoffrey E. Hinton, and Ruslan Salakhutdinov. Neighbourhood components analysis. In *NIPS*, 2004.

[10] R. Gomes, P. Welinder, A. Krause, and P. Perona. Crowdclustering. In *NIPS*, 2011.

[11] S.C.H. Hoi, W. Liu, M.R. Lyu, and W.Y. Ma. Learning distance metrics with contextual constraints for image retrieval. In *CVPR*, pages 2072–2078, 2006.

[12] Kaizhu Huang, Rong Jin, Zenglin Xu, and Cheng-Lin Liu. Robust metric learning by smooth optimization. In *UAI*, 2010.

[13] Panagiotis G. Ipeirotis. Analyzing the amazon mechanical turk marketplace. *ACM Crossroads*, 17(2):16–21, 2010.

[14] Ali Jalali, Yudong Chen, Sujay Sanghavi, and Huan Xu. Clustering partially observed graphs via convex optimization. In *ICML*, pages 1001–1008, 2011.

[15] D.C. Montgomery, E.A. Peck, and G.G. Vining. *Introduction to Linear Regression Analysis*, volume 49. John Wiley & Sons, 2007.

[16] Bernhard Scholkopf and Alexander J. Smola. *Learning with Kernels: Support Vector Machines, Regularization, Optimization, and Beyond*. MIT Press, Cambridge, MA, USA, 2001.

[17] L. Seneviratne and E. Izquierdo. Image annotation through gaming. In *Proceedings of the 2nd K-Space PhD Jamboree Workshop*, 2008.

[18] Jianbo Shi and Jitendra Malik. Normalized cuts and image segmentation. *PAMI*, 2000.

[19] Omer Tamuz, Ce Liu, Serge Belongie, Ohad Shamir, and Adam Kalai. Adaptively learning the crowd kernel. In *ICML*, 2011.

[20] K.Q. Weinberger, J. Blitzer, and L.K. Saul. Distance metric learning for large margin nearest neighbor classification. In *NIPS*, 2006.

[21] P. Welinder, S. Branson, S. Belongie, and P. Perona. The multidimensional wisdom of crowds. In *NIPS*, 2010.

[22] Lei Wu, Steven C. H. Hoi, Rong Jin, Jianke Zhu, and Nenghai Yu. Distance metric learning from uncertain side information for automated photo tagging. *ACM TIST*, 2011.

[23] E.P. Xing, A.Y. Ng, M.I. Jordan, and S. Russell. Distance metric learning, with application to clustering with side-information. In *NIPS*, 2002.

[24] Liu Yang and Rong Jin. Distance metric learning: A comprehensive survey. Technical report, Department of Computer Science and Engineering, Michigan State University, 2006.

[25] Jinfeng Yi, Rong Jin, Anil K. Jain, and Shaili Jain. Crowdclustering with sparse pairwise labels: A matrix completion approach. In *AAAI Workshop on Human Computation*, 2012.

